# TRAINING A LIMITED-INTERCONNECT, SYNTHETIC NEURAL IC

M.R. Walker, S. Haghighi, A. Afghan, and L.A. Akers
Center for Solid State Electronics Research
Arizona State University
Tempe, AZ 85287-6206
mwalker@enuxha.eas.asu.edu

## ABSTRACT

Hardware implementation of neuromorphic algorithms is hampered by high degrees of connectivity. Functionally equivalent feedforward networks may be formed by using limited fan-in nodes and additional layers, but this complicates procedures for determining weight magnitudes. No direct mapping of weights exists between fully and limited-interconnect nets. Low-level nonlinearities prevent the formation of internal representations of widely separated spatial features and the use of gradient descent methods to minimize output error is hampered by error magnitude dissipation. The judicious use of linear summations or *collection* units is proposed as a solution.

## HARDWARE IMPLEMENTATIONS OF FEEDFORWARD, SYNTHETIC NEURAL SYSTEMS

The pursuit of hardware implementations of artificial neural network models is motivated by the need to develop systems which are capable of executing neuromorphic algorithms in real time. The most significant barrier is the high degree of connectivity required between the processing elements. Current interconnect technology does not support the direct implementation of large-scale arrays of this type. In particular, the high fan-in/fan-outs of biology impose connectivity requirements such that the electronic implementation of a highly interconnected biological neural networks of just a few thousand neurons would require a level of connectivity which exceeds the current or even projected interconnection density of ULSI systems (Akers et al. 1988).

Highly layered, limited-interconnected architectures are however, especially well suited for VLSI implementations. In previous works, we analyzed the generalization and fault-tolerance characteristics of a limited-interconnect perceptron architecture applied in three simple mappings between binary input space and binary output space and proposed a CMOS architecture (Akers and Walker, 1988). This paper concentrates on developing an understanding of the limitations on layered neural network architectures imposed by hardware implementation and a proposed solution.

# TRAINING CONSIDERATIONS FOR
# LIMITED-INTERCONNECT FEEDFORWARD NETWORKS

The symbolic layout of the limited fan-in network is shown in Fig. 1. Re-arranging of the individual input components is done to eliminate edge effects. Greater detail on the actual hardware architecture may be found in (Akers and Walker, 1988) As in linear filters, the total number of connections which fan-in to a given processing element determines the degrees of freedom available for forming a hypersurface which implements the desired node output function (Widrow and Stearns, 1985). When processing elements with fixed, low fan-in are employed, the affects of reduced degrees of freedom must be considered in order to develop workable training methods which permit generalization of novel inputs. First, no direct or indirect relation exists between weight magnitudes obtained for a limited-interconnect, multilayered perceptron, and those obtained for the fully connected case. Networks of these types adapted with identical exemplar sets must therefore form completely different functions on the input space. Second, low-level nonlinearities prevent direct internal coding of widely separated spatial features in the input set. A related problem arises when hyperplane nonlinearities are used. Multiple hyperplanes required on a subset of input space are impossible when no two second level nodes address identical positions in the input space. Finally, adaptation methods like backpropagation which minimize output error with gradient descent are hindered since the magnitude of the error is dissipated as it back-propagates through large numbers of hidden layers. The appropriate placement of linear summation elements or *collection* units is a proposed solution.

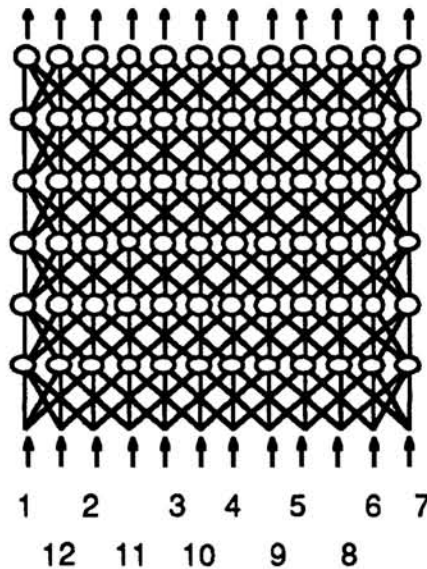

**Figure 1.** Symbolic Layout of Limited-Interconnect Feedforward Architecture

## COMPARISON OF WEIGHT VALUES IN FULLY CONNECTED AND LIMITED-INTERCONNECT NETWORKS

Fully connected and limited-interconnect feedforward structures may be functionally equivalent by virtue of identical training sets, but nonlinear node discriminant functions in a fully-connected perceptron network are generally not equivalent to those in a limited-interconnect, multilayered network. This may be shown by comparing the Taylor series expansion of the discriminant functions in the vicinity of the threshold for both types and then equating terms of equivalent order. A simple limited-interconnect network is shown in Fig. 2.

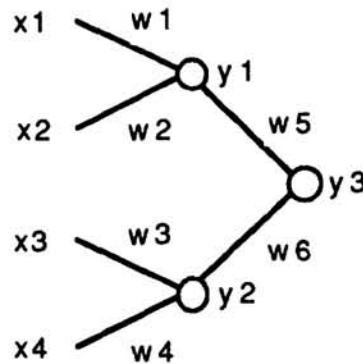

**Figure 2.** Limited-Interconnect Feedforward Network

A discriminant function with a fan-in of two may be represented with the following functional form,

$$y = f(x_a w_a + x_b w_b + \theta)$$

where $\theta$ is the threshold and the function is assumed to be continuously differentiable. The Taylor series expansion of the discriminant is,

$$y = \sum_{i=0}^{\infty} \frac{f^{(i)}(\theta)}{i!} (x_a w_a + x_b w_b)^i$$

Expanding output node three in Fig. 2 to second order,

$$y_3 = f(\theta) + f'(\theta)(y_1 w_5 + y_2 w_6) + \frac{f''(\theta)}{2}(y_1 w_5 + y_2 w_6)^2$$

where $f(\theta)$, $f'(\theta)$ and $f''(\theta)$ are constant terms. Substituting similar expansions for $y_1$ and $y_2$ into $y_3$ yields the expression,

$$y_3 = f(\theta)\left[1 + (w_5+w_6)\,f'(\theta)\right] + f'(\theta)^2\left[w_5(x_1w_1+x_2w_2) + w_6(x_3w_3+x_4w_4)\right]$$
$$+ \frac{1}{2}\,f'(\theta)\,f''(\theta)\left[w_5(x_1w_1+x_2w_2)^2 + w_6(x_3w_3+x_4w_4)^2\right]$$

The output node in the fully-connected case may also be expanded,

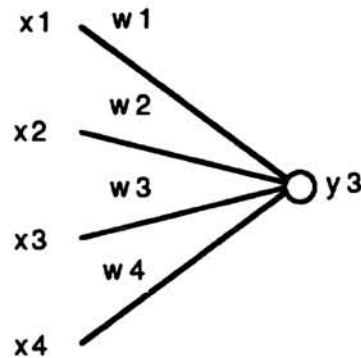

**Figure 3.** Fully Connected Network

where

$$y_3 = f(x_1w_1 + x_2w_2 + x_3w_3 + x_4w_4 + \theta)$$

Expanding to second order yields,

$$y_3 = f(\theta) + f'(\theta)\,(x_1w_1 + x_2w_2 + x_3w_3 + x_4w_4) + \frac{f''(\theta)}{2}(x_1w_1 + x_2w_2 + x_3w_3 + x_4w_4)^2$$

We seek the necessary and sufficient conditions for the two nonlinear discriminant functions to be analytically equivalent. This is accomplished by comparing terms of equal order in the expansions of each output node in the two nets. Equating the constant terms yields,

$$w_5 = -w_6$$

Equating the first order terms,

$$w_5 = w_6 = \frac{1}{f'(\theta)}$$

Equating the second order terms,

$$f'(\theta) \, w_5 w_6 - w_5 - w_6 = 0$$

The first two conditions are obviously contradictory. In addition, solving for w5 or w6 using the first and second constraints or the first and third constraints yields the trivial result, w5=w6=0. Thus, no relation exists between discriminant functions occurring in the limited and fully connected feedforward networks. This eliminates the possibility that weights obtained for a fully connected network could be transformed and used in a limited-interconnect structure. More significant is the fact that full and limited interconnect nets which are adapted with identical sets of exemplars must form completely different functions on the input space, even though they exhibit identical output behavior. For this reason, it is anticipated that the two network types could produce different responses to a novel input.

## NON-OVERLAPPING INPUT SUBSETS

Signal routing becomes important for networks in which hidden units do not address identical subsets in the proceeding layer. Figure 4 shows an odd-parity algorithm implemented with a limited-interconnect architecture. Large weight magnitudes are indicated by darker lines. Many nodes act as "pass-through" elements in that they have few dominant input and output connections. These node types are necessary to pass lower level signals to common aggregation points. In general, the use of limited fan-in processing elements implementing a nonlinear discriminant function decreases the probability that a given correlation within the input data will be encoded, especially if the "width" of the feature set is greater than the fan-in, requiring encoding at a high level within the net. In addition, since lower-level connections determine the magnitudes of upper level connections in any layered net when backpropagation is used, the set of points in weight space available to a limited-interconnect net for realizing a given function is further reduced by the greater number of weight dependencies occurring in limited-interconnect networks, all of which must be satisfied during training. Finally, since gradient descent is basically a shortcut through an NP-complete search in weight space, reduced redundancy and overlapping of internal representations reduces the probability of convergence to a near-optimal solution on the training set.

## DISSIPATION OF ERROR MAGNITUDE WITH INCREASING NUMBERS OF LAYERS

Following the derivation of backpropagation in (Plaut, 1986), the magnitude change for a weight connecting a processing element in the m-layer with a processing element in the l-layer is given by,

$$\Delta W_{m-1} \propto \frac{\partial E}{\partial W_{m-1}}$$

where

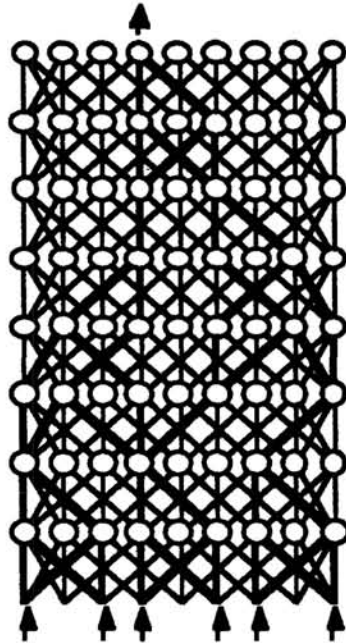

**Figure 4.** Six-input odd parity function implemented with limited-interconnect

$$\frac{\partial E}{\partial W_{m-1}} = \frac{\partial E}{\partial X_1} \frac{\partial X_1}{\partial W_{m-1}} = \frac{\partial E}{\partial Y_1} \frac{dY_1}{dX_1} \frac{\partial X_1}{\partial W_{m-1}}$$

and

$$\frac{\partial X_1}{\partial W_{m-1}} = Y_m$$

then

$$\frac{\partial E}{\partial W_{m-1}} = \left[ \sum_{k=1}^{f} \left[ \sum_{j=1}^{f} \cdots \left[ \sum_{a=1}^{f} (y_a - d_a) \frac{dy_a}{dx_a} w_{b-a} \right] \cdots \frac{dy_j}{dx_j} w_{k-j} \right] \frac{dy_k}{dx_k} w_{1-k} \right] \frac{dy_1}{dx_1} y_m$$

Where y is the output of the discriminant function, x is the activation level, w is a connection magnitude, and f is the fan-in for each processing element. If N layers of elements intervene between the m-layer and the output layer, then each of the $f^{(N-1)}$ terms in the above summation consists of the product,

$$(y_a - d_a) \frac{dy_a}{dx_a} \ w_{b-a} \cdots \frac{dy_j}{dx_j} \ w_{k-j} \ \frac{dy_k}{dx_k} \ w_{1-k} \ \frac{dy_1}{dx_1} \ y_m$$

If we replace the weight magnitudes and the derivatives in each term with their mean values,

$$(y_a - d_a) \ (\overline{\frac{dy}{dx}})^N \ (\overline{w})^{N-1} \ y_m$$

The value of the first derivative of the sigmoid discriminant function is distributed between 0.0 and 0.5. The weight values are typically initially distributed evenly between small positive and negative values. Thus with more layers, the product of the derivatives occurring in each term approaches zero. The use of large numbers of perceptron layers therefore has the affect of dissipating the magnitude of the error. This is exacerbated by the low fan-in, which reduces the total number of terms in the summation. The use of linear collection units (McClelland, 1986), discussed in the following section, is a proposed solution to this problem.

## LINEAR COLLECTION UNITS

As shown in Fig. 5, the output of the limited-interconnect net employing collection units is given by the function,

$$y_3 = f( \ w_5 c_1 (x_1 w_1 + x_2 w_2) + w_6 c_2 (x_3 w_3 + x_4 w_4) \ )$$

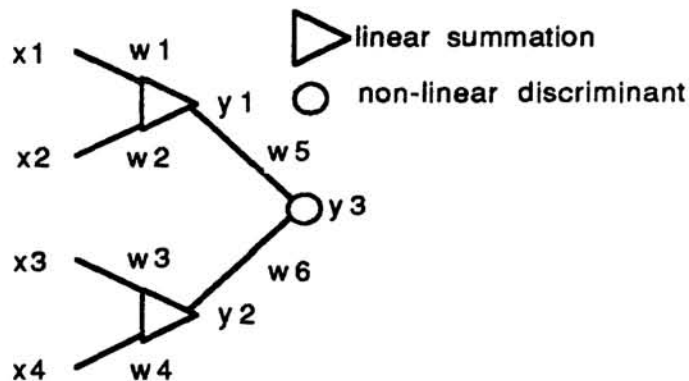

**Figure 5.** Limited-interconnect network employing linear summations

where $c_1$ and $c_2$ are constants. The position of the summations may be determined by using euclidian k-means clustering on the exemplar set to *a priori* locate cluster centers

and determine their widths (Duda and Hart, 1973). The cluster members would be combined using linear elements until they reached a nonlinear discrminant, located higher in the net and at the cluster center. With this arrangement, weights obtained for a fully-connected net could be mapped using a linear transformation into the limited-interconnect network. Alternatively, backpropagation could be used since error dissipation would be reduced by setting the linear constant c of the summation elements to arbitrarily large values.

## CONCLUSIONS

No direct transformation of weights exists between fully and limited interconnect nets which employ nonlinear discrmiminant functions. The use of gradient descent methods to minimize output error is hampered by error magnitude dissipation. In addition, low-level nonlinearities prevent the formation of internal representations of widely separated spatial features. The use of strategically placed linear summations or *collection* units is proposed as a means of overcoming difficulties in determining weight values in limited-interconnect perceptron architectures. K-means clustering is proposed as the method for determining placement.

### References

L.A. Akers, M.R. Walker, D.K. Ferry & R.O. Grondin, "Limited Interconnectivity in Synthetic Neural Systems," in R. Eckmiller and C. v.d. Malsburg eds., *Neural Computers.* Springer-Verlag, 1988.

L.A. Akers & M.R. Walker, "A Limited-Interconnect Synthetic Neural IC," *Proceedings of the IEEE International Conference on Neural Networks*, p. II-151, 1988.

B. Widrow & S.D. Stearns, *Adaptive Signal Processing.* Prentice-Hall, 1985.

D.C. Plaut, S.J. Nowlan & G.E. Hinton, "Experiments on Learning by Back Propagation," *Carnegie-Mellon University, Dept. of Computer Science Technical Report*, June, 1986.

J.L. McClelland, "Resource Requirements of Standard and Programmable Nets," in D.E. Rummelhart and J.L. McClelland eds., *Parallel Distributed Processing - Volume 1: Foundations.* MIT Press, 1986.

R.O. Duda & P.E. Hart. *Pattern Classification and Scene Analysis.* Wiley, 1973.
